# Robust Reinforcement Learning

**Jun Morimoto**
Graduate School of Information Science
Nara Institute of Science and Technology;
Kawato Dynamic Brain Project, JST
2-2 Hikaridai Seika-cho Soraku-gun
Kyoto 619-0288 JAPAN
*xmorimo@erato.atr.co.jp*

**Kenji Doya**
ATR International;
CREST, JST
2-2 Hikaridai Seika-cho Soraku-gun
Kyoto 619-0288 JAPAN
*doya@isd.atr.co.jp*

## Abstract

This paper proposes a new reinforcement learning (RL) paradigm that explicitly takes into account input disturbance as well as modeling errors. The use of environmental models in RL is quite popular for both off-line learning by simulations and for on-line action planning. However, the difference between the model and the real environment can lead to unpredictable, often unwanted results. Based on the theory of $\mathcal{H}_\infty$ control, we consider a differential game in which a 'disturbing' agent (disturber) tries to make the worst possible disturbance while a 'control' agent (actor) tries to make the best control input. The problem is formulated as finding a min-max solution of a value function that takes into account the norm of the output deviation and the norm of the disturbance. We derive on-line learning algorithms for estimating the value function and for calculating the worst disturbance and the best control in reference to the value function. We tested the paradigm, which we call "Robust Reinforcement Learning (RRL)," in the task of inverted pendulum. In the linear domain, the policy and the value function learned by the on-line algorithms coincided with those derived analytically by the linear $\mathcal{H}_\infty$ theory. For a fully nonlinear swing-up task, the control by RRL achieved robust performance against changes in the pendulum weight and friction while a standard RL control could not deal with such environmental changes.

## 1  Introduction

In this study, we propose a new reinforcement learning paradigm that we call "Robust Reinforcement Learning (RRL)." Plain, model-free reinforcement learning (RL) is desperately slow to be applied to on-line learning of real-world problems. Thus the use of environmental models have been quite common both for on-line action planning [3] and for off-line learning by simulation [4]. However, no model can

be perfect and modeling errors can cause unpredictable results, sometimes worse than with no model at all. In fact, robustness against model uncertainty has been the main subject of research in control community for the last twenty years and the result is formalized as the "$\mathcal{H}_\infty$" control theory [6].

In general, a modeling error causes a deviation of the real system state from the state predicted by the model. This can be re-interpreted as a disturbance to the model. However, the problem is that the disturbance due to a modeling error can have a strong correlation and thus standard Gaussian assumption may not be valid. The basic strategy to achieve robustness is to keep the sensitivity $\gamma$ of the feedback control loop against a disturbance input small enough so that any disturbance due to the modeling error can be suppressed if the gain of mapping from the state error to the disturbance is bounded by $1/\gamma$. In the $\mathcal{H}_\infty$ paradigm, those 'disturbance-to-error' and 'error-to-disturbance' gains are measured by a max norms of the functional mappings in order to assure stability for any modes of disturbance.

In the following, we briefly introduce the $\mathcal{H}_\infty$ paradigm and show that design of a robust controller can be achieved by finding a min-max solution of a value function, which is formulated as Hamilton-Jacobi-Isaacs (HJI) equation. We then derive on-line algorithms for estimating the value functions and for simultaneously deriving the worst disturbance and the best control that, respectively, maximizes and minimizes the value function.

We test the validity of the algorithms first in a linear inverted pendulum task. It is verified that the value function as well as the disturbance and control policies derived by the on-line algorithm coincides with the solution of Riccati equations given by $\mathcal{H}_\infty$ theory. We then compare the performance of the robust RL algorithm with a standard model-based RL in a nonlinear task of pendulum swing-up [3]. It is shown that robust RL controller can accommodate changes in the weight and the friction of the pendulum, which a standard RL controller cannot cope with.

## 2 $H_\infty$ Control

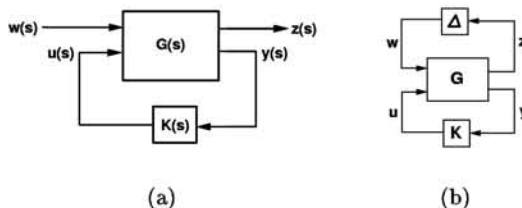

(a)                (b)

Figure 1: (a)Generalized Plant and Controller, (b)Small Gain Theorem

The standard $\mathcal{H}_\infty$ control [6] deals with a system shown in Fig.1(a), where $G$ is the plant, $K$ is the controller, $\mathbf{u}$ is the control input, $\mathbf{y}$ is the measurement available to the controller (in the following, we assume all the states are observable, i.e. $\mathbf{y} = \mathbf{x}$), $\mathbf{w}$ is unknown disturbance, and $\mathbf{z}$ is the error output that is desired to be kept small. In general, the controller $K$ is designed to stabilize the closed loop system based on a model of the plant $G$. However, when there is a discrepancy between the model and the actual plant dynamics, the feedback loop could be unstable. The effect of modeling error can be equivalently represented as a disturbance $\mathbf{w}$ generated by an

unknown mapping $\Delta$ of the plant output $\mathbf{z}$, as shown in Fig.1(b).

The goal of $\mathcal{H}_\infty$ control problem is to design a controller $K$ that brings the error $\mathbf{z}$ to zero while minimizing the $\mathcal{H}_\infty$ norm of the closed loop transfer function from the disturbance $\mathbf{w}$ to the output $\mathbf{z}$

$$\|T_{zw}\|_\infty = \sup_{\mathbf{w}} \frac{\|\mathbf{z}\|_2}{\|\mathbf{w}\|_2} = \sup_{\omega} \bar{\sigma}(T_{zw}(j\omega)). \tag{1}$$

Here $\|\bullet\|_2$ denotes $L_2$ norm and $\bar{\sigma}$ denotes maximum singular value. The small gain theorem assures that if $\|T_{zw}\|_\infty \leq \gamma$, then the system shown in Fig. 1(b) will be stable for any stable mapping $\Delta : \mathbf{z} \mapsto \mathbf{w}$ with $\|\Delta\|_\infty < \frac{1}{\gamma}$.

### 2.1 Min-max Solution to $\mathcal{H}_\infty$ Problem

We consider a dynamical system $\dot{\mathbf{x}} = f(\mathbf{x}, \mathbf{u}, \mathbf{w})$. $\mathcal{H}_\infty$ control problem is equivalent to finding a control output $\mathbf{u}$ that satisfies a constraint

$$V = \int_0^\infty (\mathbf{z}^T(t)\mathbf{z}(t) - \gamma^2 \mathbf{w}^T(t)\mathbf{w}(t))dt \leq 0 \tag{2}$$

against all possible disturbance $\mathbf{w}$ with $\mathbf{x}(0) = 0$, because it implies

$$\|T_{zw}\|_\infty^2 = \sup_{\mathbf{w}} \frac{\|\mathbf{z}\|_2^2}{\|\mathbf{w}\|_2^2} \leq \gamma^2. \tag{3}$$

We can consider this problem as differential game[5] in which the best control output $\mathbf{u}$ that minimizes $V$ is sought while the worst disturbance $\mathbf{w}$ that maximizes $V$ is chosen. Thus an optimal value function $V^*$ is defined as

$$V^* = \min_{\mathbf{u}} \max_{\mathbf{w}} \int_0^\infty (\mathbf{z}^T(t)\mathbf{z}(t) - \gamma^2 \mathbf{w}^T(t)\mathbf{w}(t))dt. \tag{4}$$

The condition for the optimal value function is given by

$$0 = \min_{\mathbf{u}} \max_{\mathbf{w}} [\mathbf{z}^T\mathbf{z} - \gamma^2 \mathbf{w}^T\mathbf{w} + \frac{\partial V^*}{\partial \mathbf{x}} f(\mathbf{x}, \mathbf{u}, \mathbf{w})] \tag{5}$$

which is known as Hamilton-Jacobi-Isaacs (HJI) equation. From (5), we can derive the optimal control output $\mathbf{u}_{op}$ and the worst disturbance $\mathbf{w}_{op}$ by solving

$$\frac{\partial \mathbf{z}^T \mathbf{z}}{\partial \mathbf{u}} + \frac{\partial V}{\partial \mathbf{x}} \frac{\partial f(\mathbf{x}, \mathbf{u}, \mathbf{w})}{\partial \mathbf{u}} = 0 \quad \text{and} \quad \frac{\partial \mathbf{z}^T \mathbf{z}}{\partial \mathbf{w}} - \gamma^2 \mathbf{w}^T + \frac{\partial V}{\partial \mathbf{x}} \frac{\partial f(\mathbf{x}, \mathbf{u}, \mathbf{w})}{\partial \mathbf{w}} = 0. \tag{6}$$

## 3 Robust Reinforcement Learning

Here we consider a continuous-time formulation of reinforcement learning [3] with the system dynamics $\dot{\mathbf{x}} = f(\mathbf{x}, \mathbf{u})$ and the reward $r(\mathbf{x}, \mathbf{u})$. The basic goal is to find a policy $\mathbf{u} = g(\mathbf{x})$ that maximizes the cumulative future reward $\int_t^\infty e^{-\frac{s-t}{\tau}} r(\mathbf{x}(s), \mathbf{u}(s))ds$ for any given state $\mathbf{x}(t)$, where $\tau$ is a time constant of evaluation. However, a particular policy that was optimized for a certain environment may perform badly when the environmental setting changes. In order to

assure robust performance under changing environment or unknown disturbance, we introduce the notion of worst disturbance in $\mathcal{H}_\infty$ control to the reinforcement learning paradigm.

In this framework, we consider an augmented reward

$$q(t) = r(\mathbf{x}(t), \mathbf{u}(t)) + s(\mathbf{w}(t)), \tag{7}$$

where $s(\mathbf{w}(t))$ is an additional reward for withstanding a disturbing input, for example, $s(\mathbf{w}) = \gamma^2 \mathbf{w}^T \mathbf{w}$. The augmented value function is then defined as

$$V(\mathbf{x}(t)) = \int_t^\infty e^{-\frac{s-t}{\tau}} q(\mathbf{x}(s), \mathbf{u}(s), \mathbf{w}(s)) ds. \tag{8}$$

The optimal value function is given by the solution of a variant of HJI equation

$$\frac{1}{\tau} V^*(\mathbf{x}) = \max_{\mathbf{u}} \min_{\mathbf{w}} [r(\mathbf{x}, \mathbf{u}) + s(\mathbf{w}) + \frac{\partial V^*}{\partial \mathbf{x}} f(\mathbf{x}, \mathbf{u}, \mathbf{w})]. \tag{9}$$

Note that we can not find appropriate policies (i.e. the solutions of the HJI equation) if we choose too small $\gamma$. In the robust reinforcement learning (RRL) paradigm, the value function is update by using the temporal difference (TD) error [3] $\delta(t) = q(t) - \frac{1}{\tau} V(t) + \dot{V}(t)$, while the best action and the worst disturbance are generated by maximizing and minimizing, respectively, the right hand side of HJI equation (9). We use a function approximator to implement the value function $V(\mathbf{x}(t); \mathbf{v})$, where $\mathbf{v}$ is a parameter vector. As in the standard continuous-time RL, we define eligibility trace for a parameter $v_i$ as $e_i(s) = \int_0^s e^{-\frac{s-t}{\kappa}} \frac{\partial V(t)}{\partial v_i} dt$ and update rule as $\dot{e}_i(t) = -\frac{1}{\kappa} e_i(t) + \frac{\partial V(t)}{\partial v_i}$, where $\kappa$ is the time constant of the eligibility trace[3]. We can then derive learning rule for value function approximator [3] as $\dot{v}_i = \eta \delta(t) e_i(t)$, where $\eta$ denotes the learning rate. Note that we do not assume $f(\mathbf{x} = 0) = 0$ because the error output $\mathbf{z}$ is generalized as the reward $r(\mathbf{x}, \mathbf{u})$ in RRL framework.

### 3.1 Actor-disturber-critic

We propose actor-disturber-critic architecture by which we can implement robust RL in a model-free fashion as the actor-critic architecture[1]. We define the policies of the actor and the disturber implemented as $\mathbf{u}(t) = A_u(\mathbf{x}(t); \mathbf{v}^u) + \mathbf{n}_u(t)$ and $\mathbf{w}(t) = A_w(\mathbf{x}(t); \mathbf{v}^w) + \mathbf{n}_w(t)$, respectively, where $A_u(\mathbf{x}(t); \mathbf{v}^u)$ and $A_w(\mathbf{x}(t); \mathbf{v}^w)$ are function approximators with parameter vectors, $\mathbf{v}^u$ and $\mathbf{v}^w$, and $\mathbf{n}_u(t)$ and $\mathbf{n}_w(t)$ are noise terms for exploration. The parameters of the actor and the disturber are updated by

$$\dot{v}_i^u = \eta^u \delta(t) \mathbf{n}_u(t) \frac{\partial A_u(\mathbf{x}(t); \mathbf{v}^u)}{\partial v_i^u} \quad \text{and} \quad \dot{v}_i^w = -\eta^w \delta(t) \mathbf{n}_w(t) \frac{\partial A_w(\mathbf{x}(t); \mathbf{v}^w)}{\partial v_i^w}, \tag{10}$$

where $\eta^u$ and $\eta^w$ denote the learning rates.

### 3.2 Robust Policy by Value Gradient

Now we assume that an input-Affine model of the system dynamics and quadratic models of the costs for the inputs are available as

$$\begin{aligned} \dot{\mathbf{x}} &= f(\mathbf{x}) + g_1(\mathbf{x})\mathbf{w} + g_2(\mathbf{x})\mathbf{u} \\ r(\mathbf{x}, \mathbf{u}) &= Q(\mathbf{x}) - \mathbf{u}^T R(\mathbf{x})\mathbf{u}, \quad s(\mathbf{w}) = \gamma^2 \mathbf{w}^T \mathbf{w}. \end{aligned}$$

In this case, we can derive the best action and the worst disturbance in reference to the value function $V$ as

$$\mathbf{u}_{op} = \frac{1}{2} R(\mathbf{x})^{-1} g_2^T(\mathbf{x}) (\frac{\partial V}{\partial \mathbf{x}})^T \quad \text{and} \quad \mathbf{w}_{op} = -\frac{1}{2\gamma^2} g_1^T(\mathbf{x}) (\frac{\partial V}{\partial \mathbf{x}})^T. \quad (11)$$

We can use the policy (11) using the value gradient $\frac{\partial V}{\partial \mathbf{x}}$ derived from the value function approximator.

### 3.3 Linear Quadratic Case

Here we consider a case in which a linear dynamic model and quadratic reward models are available as

$$\dot{\mathbf{x}} = A\mathbf{x} + B_1\mathbf{w} + B_2\mathbf{u}$$
$$r(\mathbf{x}, \mathbf{u}) = -\mathbf{x}^T Q\mathbf{x} - \mathbf{u}^T R\mathbf{u}, \quad s(\mathbf{w}) = \gamma^2 \mathbf{w}^T \mathbf{w}.$$

In this case, the value function is given by a quadratic form $V = -\mathbf{x}^T P \mathbf{x}$, where $P$ is the solution of a Riccati equation

$$A^T P + P A + P(\frac{1}{\gamma^2} B_1 B_1^T - B_2 R^{-1} B_2^T) P + Q = \frac{1}{\tau} P. \quad (12)$$

Thus we can derive the best action and the worst disturbance as

$$\mathbf{u}_{op} = R^{-1} B_2^T P \mathbf{x} \quad \text{and} \quad \mathbf{w}_{op} = -\frac{1}{\gamma^2} B_1^T P \mathbf{x}. \quad (13)$$

## 4 Simulation

We tested the robust RL algorithm in a task of swinging up a pendulum. The dynamics of the pendulum is given by $ml^2\ddot{\theta} = -\mu\dot{\theta} + mgl\sin\theta + T$, where $\theta$ is the angle from the upright position , $T$ is input torque, $\mu = 0.01$ is the coefficient of friction, $m = 1.0[\text{kg}]$ is the weight of the pendulum, $l = 1.0[\text{m}]$ is the length of the pendulum, and $g = 9.8[m/s^2]$ is the gravity acceleration. The state vector is defined as $\mathbf{x} = (\theta, \dot{\theta})^T$.

### 4.1 Linear Case

We first considered a linear problem in order to test if the value function and the policy learned by robust RL coincides with the analytic solution of $\mathcal{H}_\infty$ control problem. Thus we use a locally linearized dynamics near the unstable equilibrium point $\mathbf{x} = (0,0)^T$. The matrices for the linear model are given by

$$A = \begin{pmatrix} 0 & 1 \\ \frac{g}{l} & -\frac{\mu}{ml^2} \end{pmatrix}, B_1 = \begin{pmatrix} 0 \\ 1, \end{pmatrix}, B_2 = \begin{pmatrix} 0 \\ \frac{1}{ml^2}, \end{pmatrix}, Q = \begin{pmatrix} 1 & 0 \\ 0 & 0 \end{pmatrix}, R = 1. \quad (14)$$

The reward function is given by $q(t) = -\mathbf{x}^T Q\mathbf{x} - u^2 + \gamma^2 w^2$, where robustness criteria $\gamma = 2.0$.

The value function, $V = -\mathbf{x}^T P\mathbf{x}$, is parameterized by a symmetric matrix P. For on-line estimation of P, we define vectors $\tilde{\mathbf{x}} = (x_1^2, 2x_1x_2, x_2^2)^T$, $\mathbf{p} = (p_{11}, p_{12}, p_{22})^T$ and reformulate $V$ as $V = -\mathbf{p}^T \tilde{\mathbf{x}}$. Each element of $\mathbf{p}$ is updated using recursive least squares method[2]. Note that we used pre-designed stabilizing controller as the initial setting of RRL controller for stable learning[2].

### 4.1.1 Learning of the value function

Here we used the policy by value gradient shown in section 3.2. Figure 2(a) shows that each element of the vector $\mathbf{p}$ converged to the solution of the Ricatti equation (12).

### 4.1.2 Actor-disturber-critic

Here we used robust RL implemented by the actor-disturber-critic shown in section 3.1. In the linear case, the actor and the disturber are represented as the linear controllers, $A_u(\mathbf{x}; \mathbf{v}^u) = \mathbf{v}^u \mathbf{x}$ and $A_w(\mathbf{x}; \mathbf{v}^w) = \mathbf{v}^w \mathbf{x}$, respectively. The actor and the disturber were almost converged to the policy in (13) which derived from the Ricatti equation (12) (Fig. 2(b)).

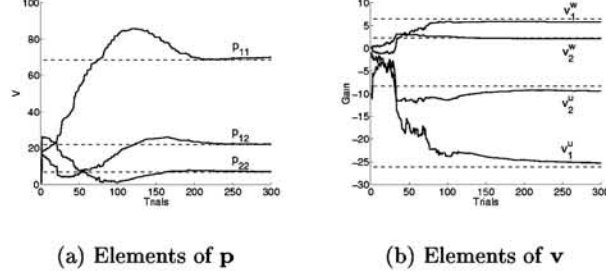

(a) Elements of $\mathbf{p}$        (b) Elements of $\mathbf{v}$

Figure 2: Time course of (a)elements of vector $\mathbf{p} = (p_{11}, p_{12}, p_{22})$ and (b)elements of gain vector of the actor $\mathbf{v}^u = (v_1^u, v_2^u)$ and the disturber $\mathbf{v}^w = (v_1^w, v_2^w)$. The dash-dotted lines show the solution of the Ricatti equation.

### 4.2 Applying Robust RL to a Non-linear Dynamics

We consider non-linear dynamical system (11), where

$$f(\mathbf{x}) = \begin{pmatrix} \dot{\theta} \\ \frac{g}{l}\sin\theta - \frac{\mu}{ml^2}\dot{\theta} \end{pmatrix}, g_1(\mathbf{x}) = \begin{pmatrix} 0 \\ 1 \end{pmatrix}, g_2(\mathbf{x}) = \begin{pmatrix} 0 \\ \frac{1}{ml^2} \end{pmatrix}$$
$$Q(\mathbf{x}) = \cos(\theta) - 1, R(\mathbf{x}) = 0.04. \tag{15}$$

From considering (7) and (15), the reward function is given by $q(t) = \cos(\theta) - 1 - 0.04u^2 + \gamma^2 w^2$, where robustness criteria $\gamma = 0.22$. For approximating the value function, we used Normalized Gaussian Network (NGnet)[3]. Note that the input gain $g(\mathbf{x})$ was also learned[3].

Fig.3 shows the value functions acquired by robust RL and standard model-based RL[3]. The value function acquired by robust RL has a shaper ridge (Fig.3(a)) attracts swing up trajectories than that learned with standard RL.

In Fig.4, we compared the robustness between the robust RL and the standard RL. Both robust RL controller and the standard RL controller learned to swing up and hold a pendulum with the weight $m = 1.0$[m] and the coefficient of friction $\mu = 0.01$ (Fig.4(a)) .

The robust RL controller could successfully swing up pendulums with different weight $m = 3.0$[kg] and the coefficient of friction $\mu = 0.3$ (Fig.4(b)). This result showed robustness of the robust RL controller. The standard RL controller could achieve the task in fewer swings for $m = 1.0$[kg] and $\mu = 0.01$ (Fig.4(a)). However, the standard RL controller could not swing up the pendulum with different weight and friction (Fig.4(b)).

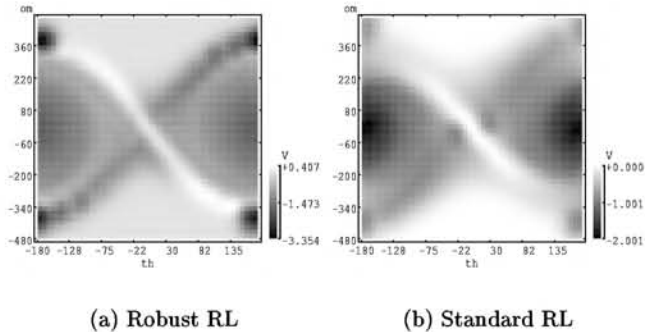

(a) Robust RL            (b) Standard RL

Figure 3: Shape of the value function after 1000 learning trials with $m = 1.0$[kg], $l = 1.0$[m], and $\mu = 0.01$.

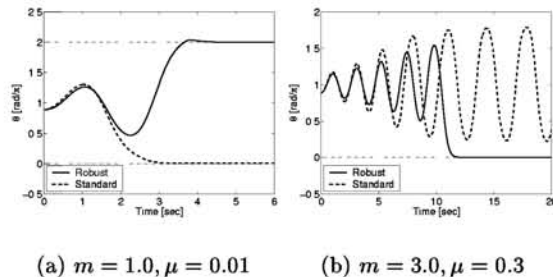

(a) $m = 1.0, \mu = 0.01$          (b) $m = 3.0, \mu = 0.3$

Figure 4: Swing up trajectories with pendulum with different weight and friction. The dash-dotted lines show upright position.

## 5    Conclusions

In this study, we proposed new RL paradigm called "Robust Reinforcement Learning (RRL)." We showed that RRL can learn analytic solution of the $\mathcal{H}_\infty$ controller in the linearized inverted pendulum dynamics and also showed that RRL can deal with modeling error which standard RL can not deal with in the non-linear inverted pendulum swing-up simulation example. We will apply RRL to more complex task like learning stand-up behavior[4].

## References

[1] A. G. Barto, R. S. Sutton, and C. W. Anderson. Neuronlike adaptive elements that can solve difficult learning control problems. *IEEE Transactions on Systems, Man, and Cybernetics*, 13:834–846, 1983.

[2] S. J. Bradtke. Reinforcement learning Applied to Linear Quadratic Regulation. In S. J. Hanson, J. D. Cowan, and C. L. Giles, editors, *Advances in Neural Information Processing Systems 5*, pages 295–302. Morgan Kaufmann, San Mateo, CA, 1993.

[3] K. Doya. Reinforcement Learning in Continuous Time and Space. *Neural Computation*, 12(1):219–245, 2000.

[4] J. Morimoto and K. Doya. Acquisition of stand-up behavior by a real robot using hierarchical reinforcement learning. In *Proceedings of Seventeenth International Conference on Machine Learning*, pages 623–630, San Francisco, CA, 2000. Morgan Kaufmann.

[5] S. Weiland. Linear Quadratic Games, $H_\infty$, and the Riccati Equation. In *Proceedings of the Workshop on the Riccati Equation in Control, Systems, and Signals*, pages 156–159. 1989.

[6] K. Zhou, J. C. Doyle, and K. Glover. *Robust Optimal Control*. PRENTICE HALL, New Jersey, 1996.
